# Integrating Topics and Syntax

**Thomas L. Griffiths**
gruffydd@mit.edu
Massachusetts Institute of Technology
Cambridge, MA 02139

**Mark Steyvers**
msteyver@uci.edu
University of California, Irvine
Irvine, CA 92614

**David M. Blei**
blei@cs.berkeley.edu
University of California, Berkeley
Berkeley, CA 94720

**Joshua B. Tenenbaum**
jbt@mit.edu
Massachusetts Institute of Technology
Cambridge, MA 02139

## Abstract

Statistical approaches to language learning typically focus on either short-range syntactic dependencies or long-range semantic dependencies between words. We present a generative model that uses both kinds of dependencies, and can be used to simultaneously find syntactic classes and semantic topics despite having no representation of syntax or semantics beyond statistical dependency. This model is competitive on tasks like part-of-speech tagging and document classification with models that exclusively use short- and long-range dependencies respectively.

## 1 Introduction

A word can appear in a sentence for two reasons: because it serves a syntactic function, or because it provides semantic content. Words that play different roles are treated differently in human language processing: function and content words produce different patterns of brain activity [1], and have different developmental trends [2]. So, how might a language learner discover the syntactic and semantic classes of words? Cognitive scientists have shown that unsupervised statistical methods can be used to identify syntactic classes [3] and to extract a representation of semantic content [4], but none of these methods captures the interaction between function and content words, or even recognizes that these roles are distinct. In this paper, we explore how statistical learning, with no prior knowledge of either syntax or semantics, can discover the difference between function and content words and simultaneously organize words into syntactic classes and semantic topics.

Our approach relies on the different kinds of dependencies between words produced by syntactic and semantic constraints. Syntactic constraints result in relatively short-range dependencies, spanning several words within the limits of a sentence. Semantic constraints result in long-range dependencies: different sentences in the same document are likely to have similar content, and use similar words. We present a model that can capture the interaction between short- and long-range dependencies. This model is a generative model for text in which a hidden Markov model (HMM) determines when to emit a word from a topic model. The different capacities of the two components of the model result in a factorization of a sentence into function words, handled by the HMM, and content words, handled by the topic model. Each component divides words into finer groups according to a different criterion: the function words are divided into syntactic classes, and the content words are

divided into semantic topics. This model can be used to extract clean syntactic and seman-
tic classes and to identify the role that words play in a document. It is also competitive in
quantitative tasks, such as part-of-speech tagging and document classification, with models
specialized to detect short- and long-range dependencies respectively.

The plan of the paper is as follows. First, we introduce the approach, considering the
general question of how syntactic and semantic generative models might be combined,
and arguing that a composite model is necessary to capture the different roles that words
can play in a document. We then define a generative model of this form, and describe
a Markov chain Monte Carlo algorithm for inference in this model. Finally, we present
results illustrating the quality of the recovered syntactic classes and semantic topics.

## 2 Combining syntactic and semantic generative models

A probabilistic generative model specifies a simple stochastic procedure by which data
might be generated, usually making reference to unobserved random variables that express
latent structure. Once defined, this procedure can be inverted using statistical inference,
computing distributions over latent variables conditioned on a dataset. Such an approach is
appropriate for modeling language, where words are generated from the latent structure of
the speaker's intentions, and is widely used in statistical natural language processing [5].

Probabilistic models of language are typically developed to capture either short-range or
long-range dependencies between words. HMMs and probabilistic context-free gram-
mars [5] generate documents purely based on syntactic relations among unobserved word
classes, while "bag-of-words" models like naive Bayes or topic models [6] generate doc-
uments based on semantic correlations between words, independent of word order. By
considering only one of the factors influencing the words that appear in documents, these
models assume that all words should be assessed on a single criterion: the posterior distri-
bution for an HMM will group nouns together, as they play the same syntactic role even
though they vary across contexts, and the posterior distribution for a topic model will assign
determiners to topics, even though they bear little semantic content.

A major advantage of generative models is modularity. A generative model for text spec-
ifies a probability distribution over words in terms of other probability distributions over
words, and different models are thus easily combined. We can produce a model that ex-
presses both the short- and long-range dependencies of words by combining two models
that are each sensitive to one kind of dependency. However, the form of combination must
be chosen carefully. In a *mixture* of syntactic and semantic models, each word would ex-
hibit either short-range or long-range dependencies, while in a *product* of models (e.g. [7]),
each word would exhibit both short-range and long-range dependencies. Consideration of
the structure of language reveals that neither of these models is appropriate. In fact, only
a subset of words – the content words – exhibit long-range semantic dependencies, while
all words obey short-range syntactic dependencies. This asymmetry can be captured in a
*composite* model, where we replace one of the probability distributions over words used in
the syntactic model with the semantic model. This allows the syntactic model to choose
when to emit a content word, and the semantic model to choose which word to emit.

### 2.1 A composite model

We will explore a simple composite model, in which the syntactic component is an HMM
and the semantic component is a topic model. The graphical model for this composite is
shown in Figure 1(a). The model is defined in terms of three sets of variables: a sequence
of words $\mathbf{w} = \{w_1, \ldots, w_n\}$, with each $w_i$ being one of $W$ words, a sequence of topic
assignments $\mathbf{z} = \{z_1, \ldots z_n\}$, with each $z_i$ being one of $T$ topics, and a sequence of
classes $\mathbf{c} = \{c_1, \ldots, c_n\}$, with each $c_i$ being one of $C$ classes. One class, say $c_i = 1$, is
designated the "semantic" class. The $z$th topic is associated with a distribution over words

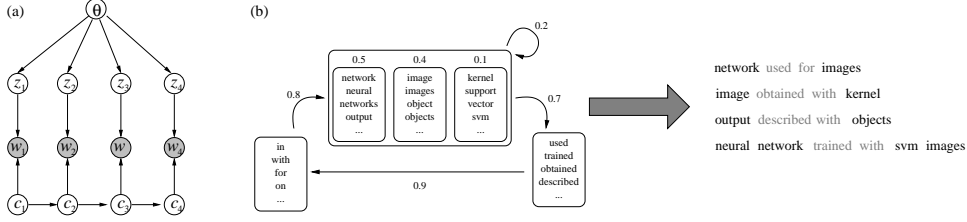

Figure 1: The composite model. (a) Graphical model. (b) Generating phrases.

$\phi^{(z)}$, each class $c \neq 1$ is associated with a distribution over words $\phi^{(c)}$, each document $d$ has a distribution over topics $\theta^{(d)}$, and transitions between classes $c_{i-1}$ and $c_i$ follow a distribution $\pi^{(s_{i-1})}$. A document is generated via the following procedure:

1. Sample $\theta^{(d)}$ from a Dirichlet($\alpha$) prior
2. For each word $w_i$ in document $d$
   (a) Draw $z_i$ from $\theta^{(d)}$
   (b) Draw $c_i$ from $\pi^{(c_{i-1})}$
   (c) If $c_i = 1$, then draw $w_i$ from $\phi^{(z_i)}$, else draw $w_i$ from $\phi^{(c_i)}$

Figure 1(b) provides an intuitive representation of how phrases are generated by the composite model. The figure shows a three class HMM. Two classes are simple multinomial distributions over words. The third is a topic model, containing three topics. Transitions between classes are shown with arrows, annotated with transition probabilities. The topics in the semantic class also have probabilities, used to choose a topic when the HMM transitions to the semantic class. Phrases are generated by following a path through the model, choosing a word from the distribution associated with each syntactic class, and a topic followed by a word from the distribution associated with that topic for the semantic class. Sentences with the same syntax but different content would be generated if the topic distribution were different. The generative model thus acts like it is playing a game of *Madlibs*: the semantic component provides a list of topical words (shown in black) which are slotted into templates generated by the syntactic component (shown in gray).

## 2.2 Inference

The EM algorithm can be applied to the graphical model shown in Figure 1, treating the document distributions $\theta$, the topics and classes $\phi$, and the transition probabilities $\pi$ as parameters. However, EM produces poor results with topic models, which have many parameters and many local maxima. Consequently, recent work has focused on approximate inference algorithms [6, 8]. We will use Markov chain Monte Carlo (MCMC; see [9]) to perform full Bayesian inference in this model, sampling from a posterior distribution over assignments of words to classes and topics.

We assume that the document-specific distributions over topics, $\theta$, are drawn from a Dirichlet($\alpha$) distribution, the topic distributions $\phi^{(z)}$ are drawn from a Dirichlet($\beta$) distribution, the rows of the transition matrix for the HMM are drawn from a Dirichlet($\gamma$) distribution, the class distributions $\phi^{(c)}$ a re drawn from a Dirichlet($\delta$) distribution, and all Dirichlet distributions are symmetric. We use Gibbs sampling to draw iteratively a topic assignment $z_i$ and class assignment $c_i$ for each word $w_i$ in the corpus (see [8, 9]).

Given the words $\mathbf{w}$, the class assignments $\mathbf{c}$, the other topic assignments $\mathbf{z}_{-i}$, and the hyperparameters, each $z_i$ is drawn from:

$$
\begin{aligned}
P(z_i|\mathbf{z}_{-i}, \mathbf{c}, \mathbf{w}) &\propto & P(z_i|\mathbf{z}_{-i}) & \quad P(w_i|\mathbf{z}, \mathbf{c}, \mathbf{w}_{-i}) \\
&\propto & \begin{cases} n_{z_i}^{(d_i)} + \alpha \\ (n_{z_i}^{(d_i)} + \alpha) \end{cases} & \quad \frac{n_{w_i}^{(z_i)}+\beta}{n_{\cdot}^{(z_i)}+W\beta} \quad \begin{matrix} c_i \neq 1 \\ c_i = 1 \end{matrix}
\end{aligned}
$$

where $n_{z_i}^{(d_i)}$ is the number of words in document $d_i$ assigned to topic $z_i$, $n_{w_i}^{(z_i)}$ is the number of words assigned to topic $z_i$ that are the same as $w_i$, and all counts include only words for which $c_i = 1$ and exclude case $i$. We have obtained these conditional distributions by using the conjugacy of the Dirichlet and multinomial distributions to integrate out the parameters $\theta, \phi$. Similarly conditioned on the other variables, each $c_i$ is drawn from:

$$P(c_i|\mathbf{c}_{-i}, \mathbf{z}, \mathbf{w}) \quad \propto \quad P(w_i|\mathbf{c}, \mathbf{z}, \mathbf{w}_{-i}) \qquad\qquad P(c_i|\mathbf{c}_{-i})$$

$$\propto \begin{cases} \dfrac{n_{w_i}^{(c_i)}+\delta}{n_{\cdot}^{(c_i)}+W\delta} \quad \dfrac{(n_{c_i}^{(c_{i-1})}+\gamma)(n_{c_{i+1}}^{(c_i)}+I(c_{i-1}=c_i)\cdot I(c_i=c_{i+1})+\gamma)}{n_{\cdot}^{(c_i)}+I(c_{i-1}=c_i)+C\gamma} & c_i \neq 1 \\[2em] \dfrac{n_{w_i}^{(z_i)}+\beta}{n_{\cdot}^{(z_i)}+W\beta} \quad \dfrac{(n_{c_i}^{(c_{i-1})}+\gamma)(n_{c_{i+1}}^{(c_i)}+I(c_{i-1}=c_i)\cdot I(c_i=c_{i+1})+\gamma)}{n_{\cdot}^{(c_i)}+I(c_{i-1}=c_i)+C\gamma} & c_i = 1 \end{cases}$$

where $n_{w_i}^{(z_i)}$ is as before, $n_{w_i}^{(c_i)}$ is the number of words assigned to class $c_i$ that are the same as $w_i$, excluding case $i$, and $n_{c_i}^{(c_{i-1})}$ is the number of transitions from class $c_{i-1}$ to class $c_i$, and all counts of transitions exclude transitions both to and from $c_i$. $I(\cdot)$ is an indicator function, taking the value 1 when its argument is true, and 0 otherwise. Increasing the order of the HMM introduces additional terms into $P(c_i|\mathbf{c}_{-i})$, but does not otherwise affect sampling.

## 3 Results

We tested the models on the Brown corpus and a concatenation of the Brown and TASA corpora. The Brown corpus [10] consists of $D = 500$ documents and $n = 1,137,466$ word tokens, with part-of-speech tags for each token. The TASA corpus is an untagged collection of educational materials consisting of $D = 37,651$ documents and $n = 12,190,931$ word tokens. Words appearing in fewer than 5 documents were replaced with an asterisk, but punctuation was included. The combined vocabulary was of size $W = 37,202$.

We dedicated one HMM class to sentence start/end markers $\{.,?,!\}$. In addition to running the composite model with $T = 200$ and $C = 20$, we examined two special cases: $T = 200$, $C = 2$, being a model where the only HMM classes are the start/end and semantic classes, and thus equivalent to Latent Dirichlet Allocation (LDA; [6]); and $T = 1$, $C = 20$, being an HMM in which the semantic class distribution does not vary across documents, and simply has a different hyperparameter from the other classes. On the Brown corpus, we ran samplers for LDA and 1st, 2nd, and 3rd order HMM and composite models, with three chains of 4000 iterations each, taking samples at a lag of 100 iterations after a burn-in of 2000 iterations. On Brown+TASA, we ran a single chain for 4000 iterations for LDA and the 3rd order HMM and composite models. We used a Gaussian Metropolis proposal to sample the hyperparameters, taking 5 draws of each hyperparameter for each Gibbs sweep.

### 3.1 Syntactic classes and semantic topics

The two components of the model are sensitive to different kinds of dependency among words. The HMM is sensitive to short-range dependencies that are constant across documents, and the topic model is sensitive to long-range dependencies that vary across documents. As a consequence, the HMM allocates words that vary across contexts to the semantic class, where they are differentiated into topics. The results of the algorithm, taken from the 4000th iteration of a 3rd order composite model on Brown+TASA, are shown in Figure 2. The model cleanly separates words that play syntactic and semantic roles, in sharp contrast to the results of the LDA model, also shown in the figure, where all words are forced into topics. The syntactic categories include prepositions, pronouns, past-tense verbs, and punctuation. While one state of the HMM, shown in the eighth column of the figure, emits common nouns, the majority of nouns are assigned to the semantic class.

The designation of words as syntactic or semantic depends upon the corpus. For comparison, we applied a 3rd order composite model with 100 topics and 50 classes to a set

| | | | | | | | | |
|---|---|---|---|---|---|---|---|---|
| the | the | the | the | the | a | the | the | the |
| blood | , | , | of | a | the | , | , | , |
| , | and | and | , | of | of | of | a | a |
| of | of | of | to | , | , | a | of | in |
| body | a | in | in | in | in | and | and | game |
| heart | in | land | in | to | water | in | drink | ball |
| and | trees | to | classes | picture | is | story | alcohol | and |
| in | tree | farmers | government | film | and | is | to | team |
| to | with | for | a | image | matter | to | bottle | to |
| is | on | farm | state | lens | are | as | in | play |

| | | | | | | | | |
|---|---|---|---|---|---|---|---|---|
| blood | forest | farmers | government | light | water | story | drugs | ball |
| heart | trees | land | state | eye | matter | stories | drug | game |
| pressure | forests | crops | federal | lens | molecules | poem | alcohol | team |
| body | land | farm | public | image | liquid | characters | people | * |
| lungs | soil | food | local | mirror | particles | poetry | drinking | baseball |
| oxygen | areas | people | act | eyes | gas | character | person | players |
| vessels | park | farming | states | glass | solid | author | effects | football |
| arteries | wildlife | wheat | national | object | substance | poems | marijuana | player |
| * | area | farms | laws | objects | temperature | life | body | field |
| breathing | rain | corn | department | lenses | changes | poet | use | basketball |
| the | in | he | * | be | said | can | time | , |
| a | for | it | new | have | made | would | way | ; |
| his | to | you | other | see | used | will | years | ( |
| this | on | they | first | make | came | could | day | : |
| their | with | i | same | do | went | may | part | ) |
| these | at | she | great | know | found | had | number | |
| your | by | we | good | get | called | must | kind | |
| her | from | there | small | go | | do | place | |
| my | as | this | little | take | | have | | |
| some | into | who | old | find | | did | | |

Figure 2: Upper: Topics extracted by the LDA model. Lower: Topics and classes from the composite model. Each column represents a single topic/class, and words appear in order of probability in that topic/class. Since some classes give almost all probability to only a few words, a list is terminated when the words account for 90% of the probability mass.

of $D = 1713$ NIPS papers from volumes 0-12. We used the full text, from the Abstract to the Acknowledgments or References section, excluding section headers. This resulted in $n = 4,312,614$ word tokens. We replaced all words appearing in fewer than 3 papers with an asterisk, leading to $W = 17,268$ types. We used the same sampling scheme as Brown+TASA. A selection of topics and classes from the 4000th iteration are shown in Figure 3. Words that might convey semantic information in another setting, such as "model", "algorithm", or "network", form part of the syntax of NIPS: the consistent use of these words across documents leads them to be incorporated into the syntactic component.

## 3.2 Identifying function and content words

Identifying function and content words requires using information about both syntactic class and semantic context. In a machine learning paper, the word "control" might be an innocuous verb, or an important part of the content of a paper. Likewise, "graph" could refer to a figure, or indicate content related to graph theory. Tagging classes might indicate that "control" appears as a verb rather than a noun, but deciding that "graph" refers to a figure requires using information about the content of the rest of the document.

The factorization of words between the HMM and LDA components provides a simple means of assessing the role that a given word plays in a document: evaluating the posterior probability of assignment to the LDA component. The results of using this procedure to identify content words in sentences excerpted from NIPS papers are shown in Figure 4. Probabilities were evaluated by averaging over assignments from all 20 samples, and take into account the semantic context of the whole document. As a result of combining short- and long-range dependencies, the model is able to pick out the words in each sentence that concern the content of the document. Selecting the words that have high probability of

| image | data | state | membrane | chip | experts | kernel | network |
|---|---|---|---|---|---|---|---|
| images | gaussian | policy | synaptic | analog | expert | support | neural |
| object | mixture | value | cell | neuron | gating | vector | networks |
| objects | likelihood | function | * | digital | hme | svm | output |
| feature | posterior | action | current | synapse | architecture | kernels | input |
| recognition | prior | reinforcement | dendritic | neural | mixture | # | training |
| views | distribution | learning | potential | hardware | learning | space | inputs |
| # | em | classes | neuron | weight | mixtures | function | weights |
| pixel | bayesian | optimal | conductance | # | function | machines | # |
| visual | parameters | * | channels | vlsi | gate | set | outputs |
| in | is | see | used | model | networks | however | # |
| with | was | show | trained | algorithm | values | also | * |
| for | has | note | obtained | system | results | then | i |
| on | becomes | consider | described | case | models | thus | x |
| from | denotes | assume | given | problem | parameters | therefore | t |
| at | being | present | found | network | units | first | n |
| using | remains | need | presented | method | data | here | - |
| into | represents | propose | defined | approach | functions | now | c |
| over | exists | describe | generated | paper | problems | hence | r |
| within | seems | suggest | shown | process | algorithms | finally | p |

Figure 3: Topics and classes from the composite model on the NIPS corpus.

1.
In contrast to this approach, we study here how the overall **network** activity can control single **cell** parameters such as **input resistance**, as well as **time** and **space** constants, parameters that are crucial for **excitability** and **spariotemporal (sic) integration**.

The integrated architecture in this paper combines **feed forward** control and **error feedback adaptive** control using **neural networks**.

2.
In other words, for our proof of **convergence**, we require the **softassign algorithm** to return a **doubly stochastic matrix** as *sinkhorn theorem guarantees that it will instead of a **matrix** which is merely close to being **doubly stochastic** based on some reasonable **metric**.

The aim is to construct a **portfolio** with a maximal **expected** return for a given **risk level** and **time horizon** while simultaneously obeying *institutional or *legally required constraints.

3.
The left graph is the standard experiment the right from a **training** with # **samples**.

The graph $G$ is called the *guest graph, and $H$ is called the host graph.

Figure 4: Function and content words in the NIPS corpus. Graylevel indicates posterior probability of assignment to LDA component, with black being highest. The boxed word appears as a function word and a content word in one element of each pair of sentences. Asterisked words had low frequency, and were treated as a single word type by the model.

being assigned to syntactic HMM classes produces templates for writing NIPS papers, into which content words can be inserted. For example, replacing the content words that the model identifies in the second sentence with content words appropriate to the topic of the present paper, we could write: *The integrated architecture in this paper combines* simple probabilistic syntax *and* topic-based semantics *using* generative models.

### 3.3 Marginal probabilities

We assessed the marginal probability of the data under each model, $P(\mathbf{w})$, using the harmonic mean of the likelihoods over the last 2000 iterations of sampling, a standard method for evaluating Bayes factors via MCMC [11]. This probability takes into account the complexity of the models, as more complex models are penalized by integrating over a latent space with larger regions of low probability. The results are shown in Figure 5. LDA outperforms the HMM on the Brown corpus, but the HMM out-performs LDA on the larger Brown+TASA corpus. The composite model provided the best account of both corpora,

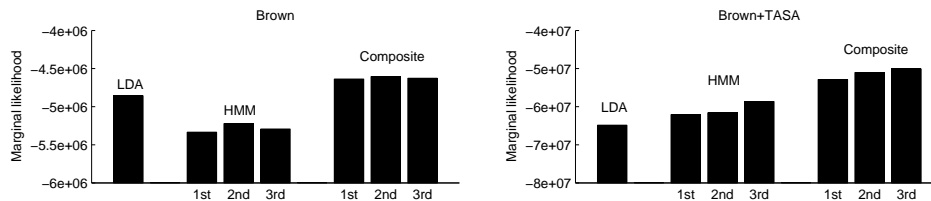

Figure 5: Log marginal probabilities of each corpus under different models. Labels on horizontal axis indicate the order of the HMM.

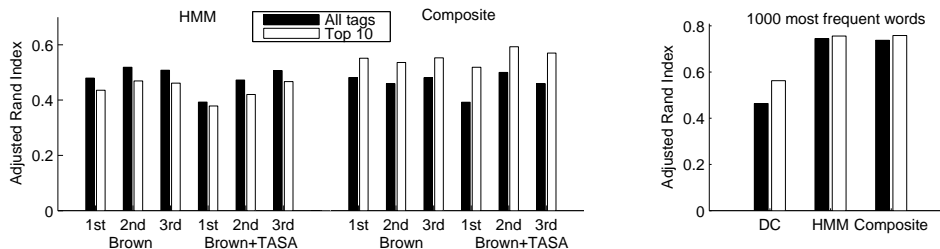

Figure 6: Part-of-speech tagging for HMM, composite, and distributional clustering (DC).

being able to use whichever kind of dependency information was most predictive. Using a higher-order transition matrix for either the HMM or the composite model produced little improvement in marginal likelihood for the Brown corpus, but the 3rd order models performed best on Brown+TASA.

### 3.4 Part-of-speech tagging

Part-of-speech tagging – identifying the syntactic class of a word – is a standard task in computational linguistics. Most unsupervised tagging methods use a lexicon that identifies the possible classes for different words. This simplifies the problem, as most words belong to a single class. However, genuinely unsupervised recovery of parts-of-speech has been used to assess statistical models of language learning, such as distributional clustering [3].

We assessed tagging performance on the Brown corpus, using two tagsets. One set consisted of all Brown tags, excluding those for sentence markers, leaving a total of 297 tags. The other set collapsed these tags into ten high-level designations: adjective, adverb, conjunction, determiner, foreign, noun, preposition, pronoun, punctuation, and verb. We evaluated tagging performance using the Adjusted Rand Index [12] to measure the concordance between the tags and the class assignments of the HMM and composite models in the 4000th iteration. The Adjusted Rand Index ranges from $-1$ to $1$, with an expectation of $0$. Results are shown in Figure 6. Both models produced class assignments that were strongly concordant with part-of-speech, although the HMM gave a slightly better match to the full tagset, and the composite model gave a closer match to the top-level tags. This is partly because all words that vary strongly in frequency across contexts get assigned to the semantic class in the composite model, so it misses some of the fine-grained distinctions expressed in the full tagset. Both the HMM and the composite model performed better than the distributional clustering method described in [3], which was used to form the 1000 most frequent words in Brown into 19 clusters. Figure 6 compares this clustering with the classes for those words from the HMM and composite models trained on Brown.

### 3.5 Document classification

The 500 documents in the Brown corpus are classified into 15 groups, such as editorial journalism and romance fiction. We assessed the quality of the topics recovered by the LDA

and composite models by training a naive Bayes classifier on the topic vectors produced by the two models. We computed classification accuracy using 10-fold cross validation for the 4000th iteration from a single chain. The two models perform similarly. Baseline accuracy, choosing classes according to the prior, was $0.09$. Trained on Brown, the LDA model gave a mean accuracy of $0.51(0.07)$, where the number in parentheses is the standard error. The 1st, 2nd, and 3rd order composite models gave $0.45(0.07), 0.41(0.07), 0.42(0.08)$ respectively. Trained on Brown+TASA, the LDA model gave $0.54(0.04)$, while the 1st. 2nd, and 3rd order composite models gave $0.48(0.06), 0.48(0.05), 0.46(0.08)$ respectively. The slightly lower accuracy of the composite model may result from having fewer data in which to find correlations: it only sees the words allocated to the semantic component, which account for approximately 20% of the words in the corpus.

## 4   Conclusion

The composite model we have described captures the interaction between short- and long-range dependencies between words. As a consequence, the posterior distribution over the latent variables in this model picks out syntactic classes and semantic topics and identifies the role that words play in documents. The model is competitive in part-of-speech tagging and classification with models that specialize in short- and long-range dependencies respectively. Clearly, such a model does not do justice to the depth of syntactic or semantic structure, or their interaction. However, it illustrates how a sensitivity to different kinds of statistical dependency might be sufficient for the first stages of language acquisition, discovering the syntactic and semantic building blocks that form the basis for learning more sophisticated representations.

**Acknowledgements.** The TASA corpus appears courtesy of Tom Landauer and Touchstone Applied Science Associates, and the NIPS corpus was provided by Sam Roweis. This work was supported by the DARPA CALO program and NTT Communication Science Laboratories.

## References

[1] H. J. Neville, D. L. Mills, and D. S. Lawson. Fractionating language: Different neural subsytems with different sensitive periods. *Cerebral Cortex*, 2:244–258, 1992.

[2] R. Brown. *A first language*. Harvard University Press, Cambridge, MA, 1973.

[3] M. Redington, N. Chater, and S. Finch. Distributional information: A powerful cue for acquiring syntactic categories. *Cognitive Science*, 22:425–469, 1998.

[4] T. K. Landauer and S. T. Dumais. A solution to Plato's problem: the Latent Semantic Analysis theory of acquisition, induction, and representation of knowledge. *Psychological Review*, 104:211–240, 1997.

[5] C. Manning and H. Schütze. *Foundations of statistical natural language processing*. MIT Press, Cambridge, MA, 1999.

[6] D. M. Blei, A. Y. Ng, and M. I. Jordan. Latent Dirichlet Allocation. *Journal of Machine Learning Research*, 3:993–1022, 2003.

[7] N. Coccaro and D. Jurafsky. Towards better integration of semantic predictors in statistical language modeling. In *Proceedings of ICSLP-98*, volume 6, pages 2403–2406, 1998.

[8] T. L. Griffiths and M. Steyvers. Finding scientific topics. *Proceedings of the National Academy of Science*, 101:5228–5235, 2004.

[9] W.R. Gilks, S. Richardson, and D. J. Spiegelhalter, editors. *Markov Chain Monte Carlo in Practice*. Chapman and Hall, Suffolk, 1996.

[10] H. Kucera and W. N. Francis. *Computational analysis of present-day American English*. Brown University Press, Providence, RI, 1967.

[11] R. E. Kass and A. E. Rafferty. Bayes factors. *Journal of the American Statistical Association*, 90:773–795, 1995.

[12] L. Hubert and P. Arabie. Comparing partitions. *Journal of Classification*, 2:193–218, 1985.
